# Non-Intrusive Gaze Tracking Using Artificial Neural Networks

**Shumeet Baluja**
baluja@cs.cmu.edu
School of Computer Science
Carnegie Mellon University
Pittsburgh, PA 15213

**Dean Pomerleau**
pomerleau@cs.cmu.edu
School of Computer Science
Carnegie Mellon University
Pittsburgh, PA 15213

## Abstract

We have developed an artificial neural network based gaze tracking system which can be customized to individual users. Unlike other gaze trackers, which normally require the user to wear cumbersome headgear, or to use a chin rest to ensure head immobility, our system is entirely non-intrusive. Currently, the best intrusive gaze tracking systems are accurate to approximately 0.75 degrees. In our experiments, we have been able to achieve an accuracy of 1.5 degrees, while allowing head mobility. In this paper we present an empirical analysis of the performance of a large number of artificial neural network architectures for this task.

## 1 INTRODUCTION

The goal of gaze tracking is to determine where a subject is looking from the appearance of the subject's eye. The interest in gaze tracking exists because of the large number of potential applications. Three of the most common uses of a gaze tracker are as an alternative to the mouse as an input modality [Ware & Mikaelian, 1987], as an analysis tool for human-computer interaction (HCI) studies [Nodine et. al, 1992], and as an aid for the handicapped [Ware & Mikaelian, 1987].

Viewed in the context of machine vision, successful gaze tracking requires techniques to handle imprecise data, noisy images, and a potentially infinitely large image set. The most accurate gaze tracking has come from intrusive systems. These systems either use devices such as chin rests to restrict head motion, or require the user to wear cumbersome equipment, ranging from special contact lenses to a camera placed on the user's head. The system described here attempts to perform non-intrusive gaze tracking, in which the user is neither required to wear any special equipment, nor required to keep his/her head still.

## 2   GAZE TRACKING

### 2.1 TRADITIONAL GAZE TRACKING

In standard gaze trackers, an image of the eye is processed in three basic steps. First, the specular reflection of a stationary light source is found in the eye's image. Second, the pupil's center is found. Finally, the relative position of the light's reflection to the pupil's center is calculated. The gaze direction is determined from information about the relative positions, as shown in Figure 1. In many of the current gaze tracker systems, the user is required to remain motionless, or wear special headgear to maintain a constant offset between the position of the camera and the eye.

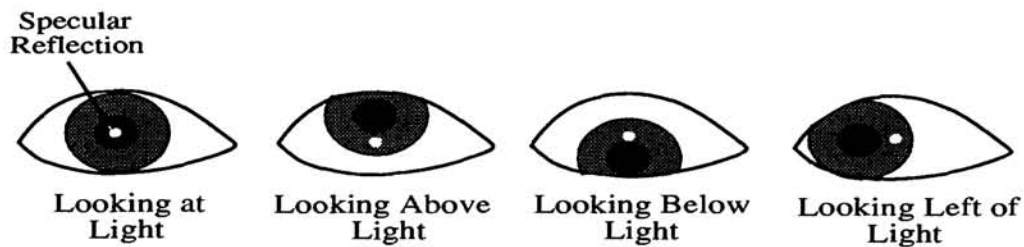

**Figure 1:** Relative position of specular reflection and pupil. This diagram assumes that the light is placed in the same location as the observer (or camera).

### 2.2 ARTIFICIAL NEURAL NETWORK BASED GAZE TRACKING

One of the primary benefits of an artificial neural network based gaze tracker is that it is non-intrusive; the user is allowed to move his head freely. In order to account for the shifts in the relative positions of the camera and the eye, the eye must be located in each image frame. In the current system, the right eye is located by searching for the specular reflection of a stationary light in the image of the user's face. This can usually be distinguished by a small bright region surrounded by a very dark region. The reflection's location is used to limit the search for the eye in the next frame. A window surrounding the reflection is extracted; the image of the eye is located within this window.

To determine the coordinates of the point the user is looking at, the pixels of the extracted window are used as the inputs to the artificial neural network. The forward pass is simulated in the ANN, and the coordinates of the gaze are determined by reading the output units. The output units are organized with 50 output units for specifying the X coordinate, and 50 units for the Y coordinate. A gaussian output representation, similar to that used in the ALVINN autonomous road following system [Pomerleau, 1993], is used for the X and Y axis output units. Gaussian encoding represents the network's response by a Gaussian shaped activation peak in a vector of output units. The position of the peak within the vector represents the gaze location along either the X or Y axis. The number of hidden units and the structure of the hidden layer necessary for this task are explored in section 3.

The training data is collected by instructing the user to visually track a moving cursor. The cursor moves in a predefined path. The image of the eye is digitized, and paired with the (X,Y) coordinates of the cursor. A total of 2000 image/position pairs are gathered. All of the networks described in this paper are trained with the same parameters for 260 epochs, using standard error back propagation. The training procedure is described in greater

detail in the next section.

# 3   THE ARTIFICIAL NEURAL NETWORK IMPLEMENTATION

In designing a gaze tracker, the most important attributes are accuracy and speed. The need for balancing these attributes arises in deciding the number of connections in the ANN, the number of hidden units needed, and the resolution of the input image. This section describes several architectures tested, and their respective performances.

## 3.1 EXAMINING ONLY THE PUPIL AND CORNEA

Many of the traditional gaze trackers look only at a high resolution picture of the subject's pupil and cornea. Although we use low resolution images, our first attempt also only used an image of the pupil and cornea as the input to the ANN. Some typical input images are shown below, in Figure 2(a). The size of the images is 15x15 pixels. The ANN architecture used is shown in Figure 2(b). This architecture was used with varying numbers of hidden units in the single, divided, hidden layer; experiments with 10, 16 and 20 hidden units were performed.

As mentioned before, 2000 image/position pairs were gathered for training. The cursor automatically moved in a zig-zag motion horizontally across the screen, while the user visually tracked the cursor. In addition, 2000 image/position pairs were also gathered for testing. These pairs were gathered while the user tracked the cursor as it followed a vertical zig-zag path across the screen. The results reported in this paper, unless noted otherwise, were all measured on the 2000 testing points. The results for training the ANN on the three architectures mentioned above as a function of epochs is shown in Figure 3. Each line in Figure 3 represents the average of three ANN training trials (with random initial weights) for each of the two users tested.

Using this system, we were able to reduce the average error to approximately 2.1 degrees, which corresponds to 0.6 inches at a comfortable sitting distance of approximately 17 inches. In addition to these initial attempts, we have also attempted to use the position of the cornea within the eye socket to aid in making finer discriminations. These experiments are described in the next section.

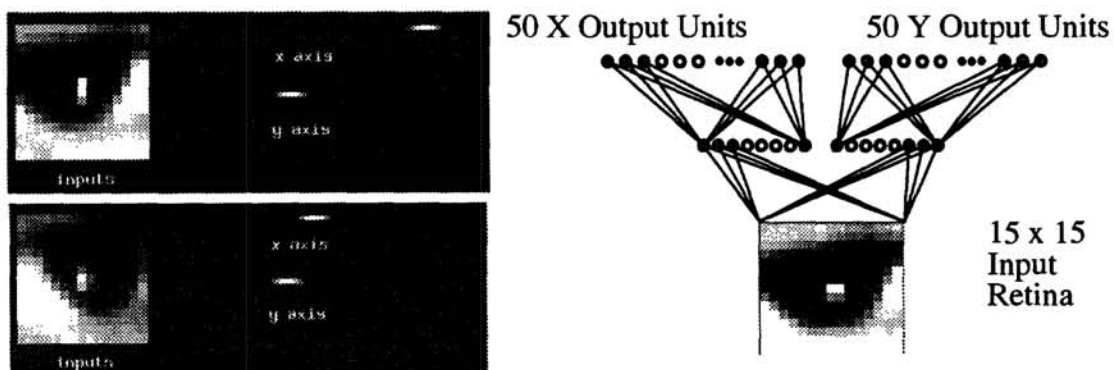

**Figure 2:** (a-left) 15 x 15 Input to the ANN. Target outputs also shown. (b-right) the ANN architecture used. A single divided hidden layer is used.

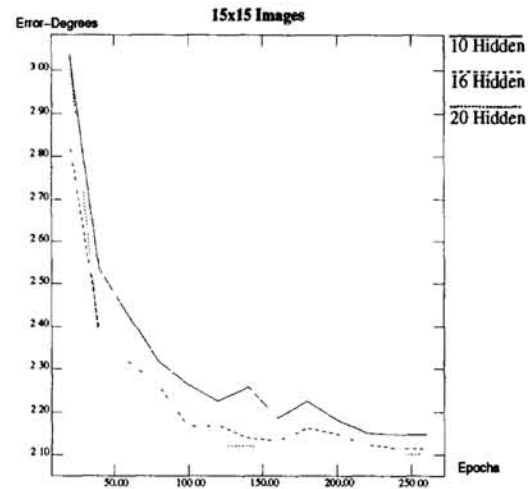

**Figure 3:** Error vs. Epochs for the 15x15 images. Errors shown for the 2000 image test set. Each line represents three ANN trainings per user; two users are tested.

## 3.2 USING THE EYE SOCKET FOR ADDITIONAL INFORMATION

In addition to using the information present from the pupil and cornea, it is possible to gain information about the subject's gaze by analyzing the position of the pupil and cornea within the eye socket. Two sets of experiments were performed using the expanded eye image. The first set used the network described in the next section. The second set of experiments used the same architecture shown in Figure 2(b), with a larger input image size. A sample image used for training is shown below, in Figure 4.

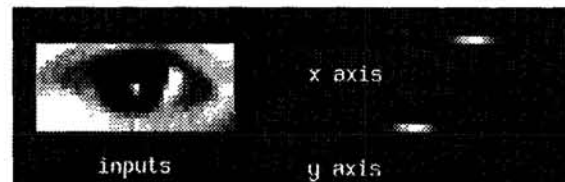

**Figure 4:** Image of the pupil and the eye socket, and the corresponding target outputs. 15 x 40 input image shown.

### 3.2.1. Using a Single Continuous Hidden Layer

One of the remaining issues in creating the ANN to be used for analyzing the position of the gaze is the structure of the hidden unit layer. In this study, we have limited our exploration of ANN architectures to simple 3 layer feed-forward networks. In the previous architecture (using 15 x 15 images) the hidden layer was divided into 2 separate parts, one for predicting the x-axis, and the other for the y-axis. Selecting this architecture over a fully connected hidden layer makes the assumption that the features needed for accurate prediction of the x-axis are not related to the features needed for predicting the y-axis. In this section, this assumption is tested. This section explores a network architecture in which the hidden layer is fully connected to the inputs and the outputs.

In addition to deciding the architecture of the ANN, it is necessary to decide on the size of the input images. Several input sizes were attempted, 15x30, 15x40 and 20x40. Surprisingly, the 20x40 input image did not provide the most accuracy. Rather, it was the 15x40 image which gave the best results. Figure 5 provides two charts showing the performance of the 15x40 and 20x40 image sizes as a function of the number of hidden units and epochs. The 15x30 graph is not shown due to space restrictions, it can be found in [Baluja & Pomerleau, 1994]. The accuracy achieved by using the eye socket information, for the 15x40 input images, is better than using only the pupil and cornea; in particular, the 15x40 input retina worked better than both the 15x30 and 20x40.

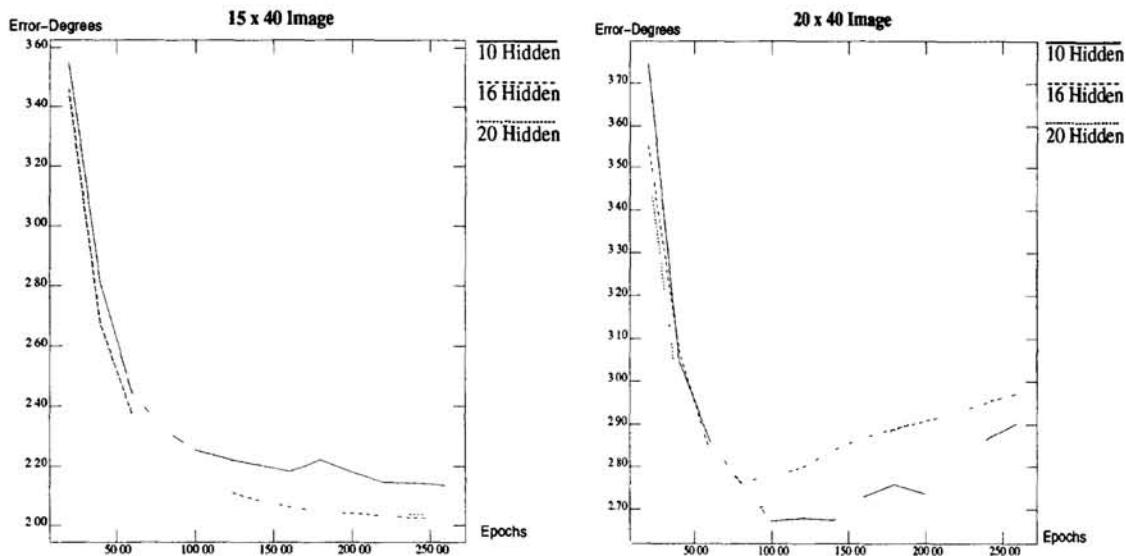

**Figure 5:** Performance of 15x40, and 20x40 input image sizes as a function of epochs and number of hidden units. Each line is the average of 3 runs. Data points taken every 20 epochs, between 20 and 260 epochs.

### 3.2.2. Using a Divided Hidden Layer

The final set of experiments which were performed were with 15x40 input images and 3 different hidden unit architectures: 5x2, 8x2 and 10x2. The hidden unit layer was divided in the manner described in the first network, shown in Figure 2(b). Two experiments were performed, with the only difference between experiments being the selection of training and testing images. The first experiment was similar to the experiments described previously. The training and testing images were collected in two different sessions, one in which the user visually tracked the cursor as it moved horizontally across the screen and the other in which the cursor moved vertically across the screen. The training of the ANN was on the "horizontally" collected images, and the testing of the network was on the "vertically" collected images. In the second experiment, a random sample of 1000 images from the horizontally collected images and a random sample of 1000 vertically collected images were used as the training set. The remaining 2000 images from both sets were used as the testing set. The second method yielded reduced tracking errors. If the images from only one session were used, the network was not trained to accurately predict gaze position independently of head position. As the two sets of data were collected in two separate sessions, the head positions from one session to the other would have changed slightly. Therefore, using both sets should have helped the network in two ways. First, the presentation of different head positions and different head movements should have improved the ability of the network to generalize. Secondly, the network was tested on images which were gathered from the same sessions as it was trained. The use of mixed training and testing sets will be explored in more detail in section 3.2.3.

The results of the first and second experiments are presented here, see Figure 6. In order to compare this architecture with the previous architectures mentioned, it should be noted that the performance of this architecture, with 10 hidden units, more accurately predicted gaze location than the architecture mentioned in section 3.2.1, in which a single continuous hidden layer was used. In comparing the performance of the architectures with 16 and 20 hidden units, the performances were very similar. Another valuable feature of using the

divided hidden layer is the reduced number of connections decreases the training and simulation times. This architecture operates at approximately 15hz. with 10 and 16 hidden units, and slightly slower with 20 hidden units.

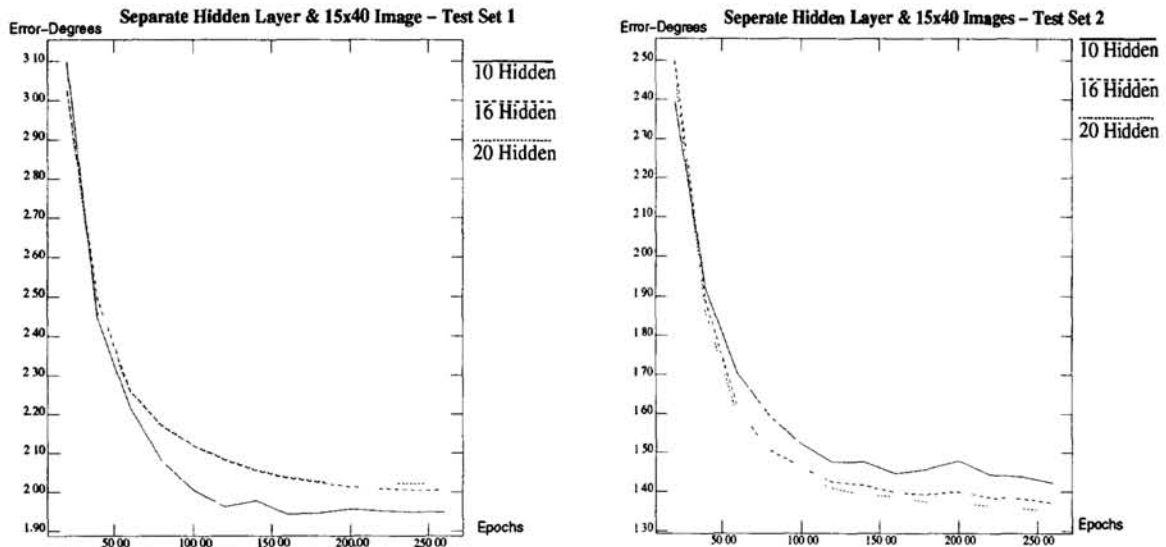

**Figure 6:** (Left) The average of 2 users with the 15x40 images, and a divided hidden layer architecture, using test setup #1. (Right) The average performance tested on 5 users, with test setup #2. Each line represents the average of three ANN trainings per user per hidden unit architecture.

### 3.2.3. Mixed Training and Testing Sets

It was hypothesized, above, that there are two reasons for the improved performance of a mixed training and testing set. First, the network ability to generalize is improved, as it is trained with more than a single head position. Second, the network is tested on images which are similar, with respect to head position, as those on which it was trained. In this section, the first hypothesized benefit is examined in greater detail using the experiments described below.

Four sets of 2000 images were collected. In each set, the user had a different head position with respect to the camera. The first two sets were collected as previously described. The first set of 2000 images (horizontal train set 1) was collected by visually tracking the cursor as it made a horizontal path across the screen. The second set (vertical test set 1) was collected by visually tracking the cursor as it moved in a vertical path across the screen. For the third and fourth image sets, the camera was moved, and the user was seated in a different location with respect to the screen than during the collection of the first training and testing sets. The third set (horizontal train set 2) was again gathered from tracking the cursor's horizontal path, while the fourth (vertical test set 2) was from the vertical path of the cursor.

Three tests were performed. In the first test, the ANN was trained using only the 2000 images in horizontal training set 1. In the second test, the network was trained using the 2000 images in horizontal training set 2. In the third test, the network was trained with a random selection of 1000 images from horizontal training set 1, and a random selection of 1000 images of horizontal training set 2. The performance of these networks was tested on both of the vertical test sets. The results are reported below, in Figure 7. The last experiment, in which samples were taken from both training sets, provides more accurate results

when testing on vertical test set 1, than the network trained alone on horizontal training set 1. When testing on vertical test set 2, the combined network performs almost as well as the network trained only on horizontal training set 2.

These three experiments provide evidence for the network's increased ability to generalize if sets of images which contain multiple head positions are used for training. These experiments also show the sensitivity of the gaze tracker to movements in the camera; if the camera is moved between training and testing, the errors in simulation will be large.

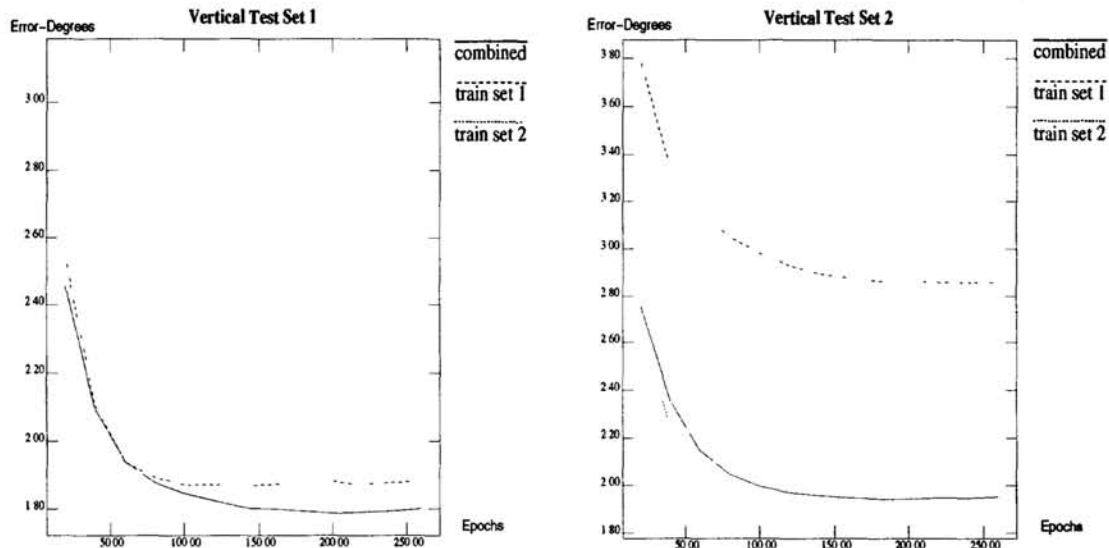

**Figure 7:** Comparing the performance between networks trained with only one head position (horizontal train set 1 & 2), and a network trained with both.

## 4  USING THE GAZE TRACKER

The experiments described to this point have used static test sets which are gathered over a period of several minutes, and then stored for repeated use. Using the same test set has been valuable in gauging the performance of different ANN architectures. However, a useful gaze tracker must produce accurate on-line estimates of gaze location. The use of an "offset table" can increase the accuracy of on-line gaze prediction. The offset table is a table of corrections to the output made by a gaze tracker. The network's gaze predictions for each image are hashed into the 2D offset-table, which performs an additive correction to the network's prediction. The offset table is filled after the network is fully trained. The user manually moves and visually tracks the cursor to regions in which the ANN is not performing accurately. The offset table is updated by subtracting the predicted position of the cursor from the actual position. This procedure can also be automated, with the cursor moving in a similar manner to the procedure used for gathering testing and training images. However, manually moving the cursor can help to concentrate effort on areas where the ANN is not performing well; thereby reducing the time required for offset table creation.

With the use of the offset table, the current system works at approximately 15 hz. The best on-line accuracy we have achieved is 1.5 degrees. Although we have not yet matched the best gaze tracking systems, which have achieved approximately 0.75 degree accuracy, our system is non-intrusive, and does not require the expensive hardware which many other systems require. We have used the gaze tracker in several forms; we have used it as an

input modality to replace the mouse, as a method of selecting windows in an X-Window environment, and as a tool to report gaze direction, for human-computer interaction studies.

The gaze tracker is currently trained for 260 epochs, using standard back propagation. Training the 8x2 hidden layer network using the 15x40 input retina, with 2000 images, takes approximately 30-40 minutes on a Sun SPARC 10 machine.

## 5   CONCLUSIONS

We have created a non-intrusive gaze tracking system which is based upon a simple ANN. Unlike other gaze-tracking systems which employ more traditional vision techniques, such as a edge detection and circle fitting, this system develops its own features for successfully completing the task. The system's average on-line accuracy is 1.7 degrees. It has successfully been used in HCI studies and as an input device. Potential extensions to the system, to achieve head-position and user independence, are presented in [Baluja & Pomerleau, 1994].

**Acknowledgments**

The authors would like to gratefully acknowledge the help of Kaari Flagstad, Tammy Carter, Greg Nelson, and Ulrike Harke for letting us scrutinize their eyes, and being "willing" subjects. Profuse thanks are also due to Henry Rowley for aid in revising this paper.

Shumeet Baluja is supported by a National Science Foundation Graduate Fellowship. This research was supported by the Department of the Navy, Office of Naval Research under Grant No. N00014-93-1-0806. The views and conclusions contained in this document are those of the authors and should not be interpreted as representing the official policies, either expressed or implied, of the National Science Foundation, ONR, or the U.S. government.

**References**

Baluja, S. Pomerleau, D.A. (1994) "Non-Intrusive Gaze Tracking Using Artificial Neural Networks" CMU-CS-94.

Jochem, T.M., D.A. Pomerleau, C.E. Thorpe (1993), "MANIAC: A Next Generation Neurally Based Autonomous Road Follower". In *Proceedings of the International Conference on Intelligent Autonomous Systems* (IAS-3).

Nodine, C.F., H.L. Kundel, L.C. Toto & E.A. Krupinksi (1992) "Recording and analyzing eye-position data using a microcomputer workstation", *Behavior Research Methods, Instruments & Computers* 24 (3) 475-584.

Pomerleau, D.A. (1991) "Efficient Training of Artificial Neural Networks for Autonomous Navigation," *Neural Computation 3*:1, Terrence Sejnowski (Ed).

Pomerleau, D.A. (1993) *Neural Network Perception for Mobile Robot Guidance*. Kluwer Academic Publishing.

Pomerleau, D.A. (1993) "Input Reconstruction Reliability Estimation", *Neural Information Processing Systems 5*. Hanson, Cowan, Giles (eds.) Morgan Kaufmann, pp. 270-286.

Starker, I. & R. Bolt (1990) "A Gaze-Responsive Self Disclosing Display", In *CHI-90*. Addison Wesley, Seattle, Washington.

Waibel, A., Sawai, H. & Shikano, K. (1990) "Consonant Recognition by Modular Construction of Large Phonemic Time-Delay Neural Networks". *Readings in Speech Recognition*. Waibel and Lee.

Ware, C. & Mikaelian, H. (1987) "An Evaluation of an Eye Tracker as a Device for Computer Input", In J. Carrol and P. Tanner (ed.) *Human Factors in Computing Systems - IV.* Elsevier.